# Back Propagation Implementation on the Adaptive Solutions CNAPS Neurocomputer Chip

**Hal McCartor**
Adaptive Solutions Inc.
1400 N.W. Compton Drive
Suite 340
Beaverton, OR 97006

## Abstract

The Adaptive Solutions CNAPS architecture chip is a general purpose neurocomputer chip. It has 64 processors, each with 4 K bytes of local memory, running at 25 megahertz. It is capable of implementing most current neural network algorithms with on chip learning. This paper discusses the implementation of the Back Propagation algorithm on an array of these chips and shows performance figures from a clock accurate hardware simulator. An eight chip configuration on one board can update 2.3 billion connections per second in learning mode and process 9.6 billion connections per second in feed forward mode.

## 1   Introduction

The huge computational requirements of neural networks and their natural parallelism have led to a number of interesting hardware innovations for executing such networks. Most investigators have created large parallel computers or special purpose chips limited to a small subset of algorithms. The Adaptive Solutions CNAPS architecture describes a general-purpose 64-processor chip which supports on chip learning and is capable of implementing most current algorithms. Implementation of the popular Back Propagation (BP) algorithm will demonstrate the speed and versatility of this new chip.

## 2   The Hardware Resources

The Adaptive Solutions CNAPS architecture is embodied in a single chip digital neurocomputer with 64 processors running at 25 megahertz. All processors receive the same instruction which they conditionally execute. Multiplication and addition are performed in parallel allowing 1.6 billion inner product steps per second per chip. Each processor has a 32-bit adder, 9-bit by 16-bit multiplier (16 by 16 in two clock cycles), shifter, logic unit, 32 16-bit registers, and 4096 bytes of local memory. Input and output are accomplished over 8-bit input and output buses common to all processors. The output bus is tied to the input bus so that output of one processor can be broadcast to all others. When multiple chips are used, they appear to the user as one chip with more processors. Special circuits support finding the maximum of values held in each processor and conserving weight space for sparsely connected networks. An accompanying sequencer chip controls instruction flow, input and output.

## 3   The Back Propagation Algorithm Implementation

Three critical issues must be addressed in the parallel implementation of BP on efficient hardware. These are the availability of weight values for back propagating the error, the scaling and precision of computations, and the efficient implementation of the output transfer function.

BP requires weight values at different nodes during the feed forward and back propagation phases of computation. This problem is solved by having a second set of weights which is the transpose of the output layer weights. These are located on hidden node processors. The two matrices are updated identically. The input to the hidden layer weight matrix is not used for error propagation and is not duplicated.

BP implementations typically use 32-bit floating point math. This largely eliminates scaling, precision and dynamic range issues. Efficient hardware implementation dictates integer arithmetic units with precision no greater than required. Baker [Bak90] has shown 16-bit integer weights are sufficient for BP training and much lower values adequate for use after training.

With fixed point integer math, the position of the binary point must be chosen. In this implementation weights are 16 bits and use 12 bits to the right of the binary point and four to the left including a sign bit. They range from -8 to +8. Input and output are represented as 8-bit unsigned integers with binary point at the left. The leaning rate is represented as an 8-bits integer with two bits to the left of the binary point and values ranging from .016 to 3.98. Error is represented as 8 bit signed integers at the output layer and with the same representation as the weights at the hidden layer.

This data representation has been used to train benchmark BP applications with results comparable to the floating point versions [HB91].

The BP sigmoid output function is implemented as an 8-bit by 256 lookup table.

During the forward pass input values are broadcast to all processors from off chip via the input bus or from hidden nodes via the output bus to the input bus. During

the backward error propagation, error values are broadcast from the output nodes to hidden nodes.

The typical BP network has two computational layers, the hidden and output layers. They can be assigned to the same or different processor nodes (PNs) depending on available memory for weights. PNs used for the hidden layer contain the transpose weights of the output layer for back propagating error. If momentum or periodic weight update are used, additional storage space is allocated with each weight.

In this implementation BP can be mapped to any set of contiguous processors allowing multiple networks in CNAPS memory simultaneously. Thus, the output of one algorithm can be directly used as input to another. For instance, in speech recognition, a Fourier transform performed on the PN array could be input to a series of matched BP networks whose hidden layers run concurrently. Their output could be directed to an LVQ2 network for final classification. This can all be accomplished without any intermediate results leaving the chip array.

## 4  Results

BP networks have been successfully run on a hardware clock accurate simulator which gives the following timing results. In this example an eight-chip implementation (512 processors) was used. The network had 1900 inputs, 500 hidden nodes and 12 outputs. Weights were updated after each input and no momentum was used. The following calculations show BP performance:

### TRAINING PHASE

Overhead clock cycles per input vector = 360
Cycles per input vector element = 4
Cycles per hidden node = 4
Cycles per output node = 7
Cycles per vector = 360+(1900*4)+(500*4)+(12*7) = 10,044
Vectors per second = 25,000,000 / 10,044 = 2,489
Total forward weights = (1900*500)+(500*12) = 956,000

**Weight updates per second = 956,000*2,489 = 2,379,484,000**

### FEED FORWARD ONLY

Overhead cycles per input vector = 59
Cycles per input vector element = 1
Cycles per hidden node = 1
Cycles per output node = 1 (for output of data)
Cycles per vector = 59+1900+500+12 = 2,471
Vectors per second = 25,000,000/2,471 = 10,117

**Connections per second = 956,000*10,117 = 9,671,852,000**

## 5   Comparative Performance

An array of eight Adaptive Solutions CNAPS chips would execute the preceding BP network at 2.3 billion training weight updates per second or 9.6 billion feed forward connections per second. These results can be compared with the results on other computers shown in table 1.

| MACHINE | MCUPS | MCPS | WTS |
|---|---|---|---|
| SUN 3 [D88] | .034 | 0.25 | fp |
| SAIC SIGMA-1 [D88] | | 8 | fp |
| WARP [PGTK88] | 17 | | fp |
| CRAY 2 [PGTK88] | 7 | | fp |
| CRAY X-MP [D88] | | 50 | fp |
| CM-2 (65,536) [ZMMW90] | 40 | 182 | fp |
| GF-11 (566) [WZ89] | 901 | | fp |
| 8 ADAPTIVE CNAPS chips | 2,379 | 9,671 | 16 bit int |

Table 1. Comparison of BP performance for various computers and 8 Adaptive Solutions CNAPS chips on one board. MCUPS is Millions of BP connection updates per second in training mode. MCPS is millions of connections processed per second in feed forward mode. WTS is representation used for weights.

## 6   Summary

The Adaptive Solutions CNAPS chip is a very fast general purpose digital neuro-computer chip. It is capable of executing the Back Propagation algorithm quite efficiently. An 8 chip configuration can train 2.3 billion connections per second and evaluate 9.6 billion BP feed forward connections per second.

### References

[Bak90] T Baker. Implementation limits for artificial neural networks. Master's thesis, Oregon Graduate Institute, 1990.

[D88] DARPA Neural Network Study. pp309-310 AFCEA International Press, Fairfax Virginia. 1988

[HB91] J. Holt and T. Baker. Back Propagation Simulations using Limited Precision Calculations. Submitted to IJCNN, Seattle WA 1991.

[RM86] D. Rummelhart, J. McClelland. Parallel Distributed Processing. (1986) MIT Press, Cambridge, MA.

[WZ89] M. Witbrock and M Zagha. An Implementation of Back-Propagation Learning on GF11, a Large SIMD Parallel Computer. 1989. Tech report CMU-CS-89-208 Carnegie Mellon University.

[ZMMW90] X. Zhang, M. Mckenna, J Misirov, D Waltz. An Efficient Implementation of the Back-propagation Algorithm on the Connection Machine CM-2 (1990) in Adv. in Neural Information Processing Systems 2. Ed. D. Touretzky. Morgan Kaufmann, San Mateo, CA.
